# Word Features for Latent Dirichlet Allocation

**James Petterson**[1]**, Alex Smola**[2]**, Tiberio Caetano**[1]**, Wray Buntine**[1]**, Shravan Narayanamurthy**[3]
[1]NICTA and ANU, Canberra, ACT, Australia
[2]Yahoo! Research, Santa Clara, CA, USA
[3]Yahoo! Research, Bangalore, India

## Abstract

We extend Latent Dirichlet Allocation (LDA) by explicitly allowing for the encoding of side information in the distribution over words. This results in a variety of new capabilities, such as improved estimates for infrequently occurring words, as well as the ability to leverage thesauri and dictionaries in order to boost topic cohesion within and across languages. We present experiments on multi-language topic synchronisation where dictionary information is used to bias corresponding words towards similar topics. Results indicate that our model substantially improves topic cohesion when compared to the standard LDA model.

## 1 Introduction

Latent Dirichlet Allocation [4] assigns topics to documents and generates topic distributions over words given a collection of texts. In doing so, it ignores any side information about the similarity between words. Nonetheless, it achieves a surprisingly high quality of coherence within topics.

The inability to deal with word features makes LDA fall short on several aspects. The most obvious one is perhaps that the topics estimated for infrequently occurring words are usually unreliable. Ideally, for example, we would like the topics associated with *synonyms* to have a prior tendency of being similar, so that in case one of the words is rare but the other is common, the topic estimates for the rare one can be improved. There are other examples. For instance, it is quite plausible that 'Germany' and 'German', or 'politics', 'politician', and 'political' should, by default, belong to the same topic. Similarly, we would like to be able to leverage dictionaries in order to boost topic cohesion across languages, a problem that has been researched but is far from being fully solved, especially for non-aligned corpora [6]. For example, we know that 'democracy' and 'democracia' are different words, but it is clear that not leveraging the fact they actually mean the same thing (and therefore should have aligned topics) reduces the statistical strength of a model.

A possible solution, which we propose in this paper, is to treat word information as *features* rather than as explicit constraints and to adjust a smoothing prior over topic distributions for words such that correlation is emphasised. In the parlance of LDA we do *not* pick a globally constant $\beta$ smoother over the word multinomials but rather we adjust it according to word similarity. In this way we are capable of learning the prior probability of how words are distributed over various topics based on how similar they are, e.g. in the context of dictionaries, synonym collections, thesauri, edit distances, or distributional word similarity features.

Unfortunately, in performing such model extension we lose full tractability of the setting by means of a collapsed Gibbs sampler. Instead, we use a hybrid approach where we perform smooth optimisation over the word smoothing coefficients, while retaining a collapsed Gibbs sampler to assign topics for a fixed choice of smoothing coefficients. The advantage of this setting is that it is entirely modular and can be added to existing Gibbs samplers without modification.

We present experimental results on multi-language topic synchronisation which clearly evidence the ability of the model to incorporate dictionary information successfully. Using several different measures of topic alignment, we consistently observe that the proposed model improves substantially on standard LDA, which is unable to leverage this type of information.

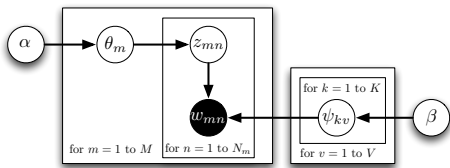

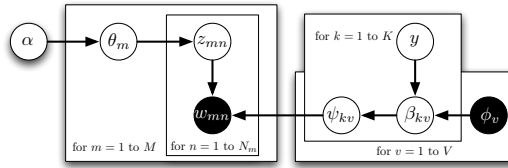

Figure 1: LDA: The topic distribution for each word ($\psi_v$) has as smoother the Dirichlet distribution with a parameter $\beta$ (independent of the word).

Figure 2: Our Extension: Assume we observe side information $\phi_v$ (i.e. features) for each word $v$. The word-specific smoothing parameters $\beta_{kv}$ are governed by $\phi_v$ and a common parameter choice $y$.

## 1.1 Related work

Loosely related works that use logistic models to induce structure in generative models are [17], which proposed a shared logistic normal distribution as a Bayesian prior over probabilistic grammar weights, and [10], which incorporated features into unsupervised models using locally normalized models. More related to our work is [5], which encodes correlations between synonyms, and [1] which encodes more general correlations. In fact, our proposed model can be seen as a generalisation of [1], where we can encode the strength of the links between each pair of words.

Previous work on multilingual topic models requires parallelism at either the sentence level ([20]) or document level ([9], [15]). More recent work [13] relaxes that, but still requires that a significant fraction (at least 25%) of the documents are paired up.

Multilingual topic alignment without parallelism was recently proposed by [6]. Their model requires a list of matched word pairs $m$ (where each pair has one word in each language) and corresponding matching priors $\pi$ that encode the prior knowledge on how likely the match is to occur. The topics are defined as distributions over word pairs, while the unmatched words come from a unigram distribution specific to each language. Although their model could be in principle extended to more than two languages their experimental section was focused on the bilingual case.

One of the key differences between [6] and our method is that we do not hardcode word information, but we use it only as a prior – this way our method becomes less sensitive to errors in the word features. Furthermore, our model automatically extends to multiple languages without any modification, aligning topics even for language pairs for which we have no information, as we show in the experimental section for the Portuguese/French pair. Finally, our model is conceptually simpler and can be incorporated as a module in existing LDA implementations.

## 2 The Model

We begin by briefly reviewing the LDA model of [4] as captured in Figure 1. It assumes that

$$\theta_m \sim \text{Dir}(\alpha) \quad (1a) \qquad \psi_k \sim \text{Dir}(\beta) \quad (1c)$$
$$z_{mn} \sim \text{Mult}(\theta_m) \quad (1b) \qquad w_{mn} \sim \text{Multi}(\psi_{z_{mn}}) \quad (1d)$$

Nonparametric extensions in terms of the number of topics can be obtained using Dirichlet process models [2] regarding the generation of topics. Our extension deals with the word smoother $\beta$. Instead of treating it as a constant for all words we attempt to infer its values for different words and topics respectively. That is, we assume that (1c) is replaced by

$$\psi_k \sim \text{Dir}(\beta_k|\phi, y) \quad (2a) \qquad \beta \sim \text{Logistic}(y; \phi). \quad (2b)$$

We refer to this setting as *downstream conditioning*, in analogy to the *upstream conditioning* of [14] (which dealt with topical side information over documents). The corresponding graphical model is given in Figure 2. The above dependency allows us to incorporate features of words as side information. For instance, if two words (e.g. 'politics' and 'politician') are very similar then it is plausible to assume that their topic distributions should also be quite similar. This can be achieved by choosing similar $\beta_{k,\text{politics}}$ and $\beta_{k,\text{politician}}$. For instance, both of those coefficients might have great affinity to $\beta_{k,\text{scandal}}$ and we might estimate $y$ such that this is achieved.

## 2.1 Detailed Description

We now discuss the directed graphical model from Figure 2 in detail. Whenever needed we use the *collapsed* representation of the model [8], that is we integrate out the parameters $\theta_m$ and $\psi_{kv}$ such that we only need to update $\alpha$ and $\beta$ (or indirectly $y$). We define the standard quantities

$$n_{kv}^{\mathrm{KV}} = \sum_{m,n} \{z_{mn} = k \text{ and } w_{mn} = v\} \qquad n_k^{\mathrm{K}} = \sum_m n_{km}^{\mathrm{KM}} \qquad\qquad n_m^{\mathrm{M}} = \sum_k n_{km}^{\mathrm{KM}}$$

$$n_{km}^{\mathrm{KM}} = \sum_n \{z_{mn} = k\} \qquad\qquad\qquad n_v^{\mathrm{V}} = \sum_k n_{kv}^{\mathrm{KV}}, \qquad\qquad \text{as well as:}$$

**Topic distribution** $p(z_{mn}|\theta_m)$: We assume that this is a multinomial distribution specific to document $m$, that is $p(z_{mn}|\theta_m) = \theta_{m,z_{mn}}$.

**Conjugate distribution** $p(\theta_m|\alpha)$: This is a Dirichlet distribution with parameters $\alpha$, where $\alpha_k$ denotes the smoother for topic $k$.

**Collapsed distribution** $p(z_m|\alpha)$: Integrating out $\theta_m$ and using conjugacy yields

$$p(z_m|\alpha) = \frac{\prod_{k=1}^K \Gamma(n_{km}^{\mathrm{KM}} + \alpha_k)}{\Gamma\left(n_m^{\mathrm{M}} + \|\alpha\|_1\right)} \frac{\Gamma\left(\|\alpha\|_1\right)}{\prod_{k=1}^K \Gamma(\alpha_k)},$$

where $\Gamma$ is the gamma function: $\Gamma(x) = \int_0^\infty t^{x-1} e^{-t}\, dt$.

**Word distribution** $p(w_{mn}|z_{mn}, \psi)$: We assume that given a topic $z_{mn}$ the word $w_{mn}$ is drawn from a multinomial distribution $\psi_{w_{mn}, z_{mn}}$. That is $p(w_{mn}|z_{mn}, \psi) = \psi_{w_{mn}, z_{mn}}$. This is entirely standard as per the basic LDA model.

**Conjugate distribution** $p(\psi_k|\beta_k)$: As by default, we assume that $\psi_k$ is distributed according to a Dirichlet distribution with parameters $\beta_k$. The key difference is that here we do not assume that all coordinates of $\beta_k$ are identical. Nor do we assume that all $\beta_k$ are the same.

**Collapsed distribution** $p(w|z, \beta)$: Integrating out $\psi_k$ for all topics $k$ yields the following

$$p(w|z, \beta) = \prod_{k=1}^K \frac{\prod_{v=1}^V \Gamma(n_{kv}^{\mathrm{KV}} + \beta_{kv})}{\Gamma\left(n_k^{\mathrm{K}} + \|\beta_k\|_1\right)} \frac{\Gamma\left(\|\beta_k\|_1\right)}{\prod_{v=1}^V \Gamma(\beta_{kv})}$$

## 2.2 Priors

In order to better control the capacity of our model, we impose a prior on naturally related words, e.g. the ('Toyota', 'Kia') and the ('Bush', 'Cheney') tuples, rather than generally related words. For this purpose we design a similarity graph $G(V, E)$ with words represented as vertices $V$ and similarity edge weights $\phi_{uv}$ between vertices $u, v \in V$ whenever $u$ is related to $v$. In particular, the magnitude of $\phi_{uv}$ can denote the similarity between words $u$ and $v$.

In the following we denote by $y_{kv}$ the topic dependent smoothing coefficients for a given word $v$ and topic $k$. We impose the smoother

$$\log \beta_{kv} = y_{kv} + y_v \text{ and } \log p(\beta) = \frac{-1}{2\lambda^2}\left[\sum_{v,v',k} \phi_{v,v'}(y_{kv} - y_{kv'})^2 + \sum_v y_v^2\right]$$

where $\log p(\beta)$ is given up to an additive constant and $y_v$ allows for multiplicative topic-unspecific corrections. A similar model was used by [3] to capture temporal dependence between topic models computed at different time instances, e.g. when dealing with topic drift over several years in a scientific journal. There the vertices are words at a given time and the edges are between smoothers instantiated at subsequent years.

## 3 Inference

In analogy to the collapsed sampler of [8] we also represent the model in a collapsed fashion. That is, we integrate out the random variables $\theta_m$ (the document topic distributions) and $\psi_{kv}$ (the topic

word distributions), which leads to a joint likelihood in terms of the actual words $w_{mn}$, the side information $\phi$ about words, the latent variable $y$, the smoothing hyperprior $\beta_{kv}$, and finally, the topic assignments $z_{mn}$.

## 3.1 Document Likelihood

The likelihood contains two terms: a word-dependent term which can be computed on the fly while resampling data[1], and a model-dependent term involving the topic counts and the word-topic counts which can be computed by one pass through the aggregate tables respectively. Let us first write out the uncollapsed likelihood in terms of $z, \theta, \psi, \alpha, \beta$. We have

$$
p(w, z, \theta, \psi | \alpha, \beta) = \prod_{m=1}^{M} \prod_{n=1}^{N_m} p(w_{mn}|z_{mn}, \psi) p(z_{mn}|\theta_m) \prod_{m=1}^{M} p(\theta_m|\alpha) \prod_{k=1}^{K} p(\psi_k|\beta)
$$

Define $\bar{\alpha} := \|\alpha\|_1$ and $\bar{\beta}_k := \|\beta_k\|_1$. Integrating out $\theta$ and $\psi$ yields

$$
p(w, z | \alpha, \beta) = \prod_{m=1}^{M} \frac{\Gamma(\bar{\alpha})}{\Gamma(\bar{\alpha} + n_m^{\mathrm{M}})} \prod_{k: n_{km}^{\mathrm{KM}} \neq 0} \frac{\Gamma(\alpha_k + n_{km}^{\mathrm{KM}})}{\Gamma(\alpha_k)} \prod_{k=1}^{K} \frac{\Gamma(\bar{\beta}_k)}{\Gamma(\bar{\beta}_k + n_k^{\mathrm{K}})} \prod_{v: n_{kv}^{\mathrm{KV}} \neq 0} \frac{\Gamma(\beta_{kv} + n_{kv}^{\mathrm{KV}})}{\Gamma(\beta_{kv})}
$$

The above product is obtained simply by canceling out terms in denominator and numerator where the counts vanish. This is computationally significant, since it allows us to evaluate the normalization for sparse count tables with cost linear in the number of nonzero coefficients rather than cost in the dense count table.

## 3.2 Collapsed Sampler

In order to perform inference we need two components: a sampler which is able to draw from $p(z_i = k | w, z_{\neg i}, \alpha, \beta)$[2], and an estimation procedure for $(\beta, y)$. The sampler is essentially the same as in standard LDA. For the count variables $n^{\mathrm{KM}}, n^{\mathrm{KV}}, n^{\mathrm{K}}$ and $n^{\mathrm{M}}$ we denote by the subscript '$-$' their values after the word $w_{mn}$ and associated topic $z_{mn}$ have been removed from the statistics. Standard calculations yield the following topic probability for resampling:

$$
p(z_{mn} = k | \mathrm{rest}) \propto \frac{\left[\beta_{kv} + n_{kv_{mn}-}^{\mathrm{KV}}\right] \left[n_{km-}^{\mathrm{KM}} + \alpha_k\right]}{n_{k-}^{\mathrm{K}} + \bar{\beta}_k} \tag{6}
$$

In the appendix we detail how to addapt the sampler of [19] to obtain faster sampling.

## 3.3 Topic Smoother for $\beta$

Optimizing over $y$ is considerably hard since the log-likelihood does not decompose efficiently. This is due to the dependence of $\bar{\beta}_k$ on all words in the dictionary. The data-dependent contribution to the negative log-likelihood is

$$
L_\beta = \sum_{k=1}^{K} \left[\log \Gamma(\bar{\beta}_k + n_k^{\mathrm{K}}) - \log \Gamma(\bar{\beta}_k)\right] + \sum_{k=1}^{K} \sum_{v: n_{kv}^{\mathrm{KV}} \neq 0} \left[\log \Gamma(\beta_{kv}) - \log \Gamma(\beta_{kv} + n_{kv}^{\mathrm{KV}})\right]
$$

with gradients given by the appropriate derivatives of the $\Gamma$ function. We use the prior from section 2.2, which smooths between closely related words only. After choosing edges $\phi_{uv}$ according to these matching words, we obtain an optimisation problem directly in terms of the variables $y_{kv}$ and $y_v$. Denote by $N(v)$ the neighbours for word $v$ in $G(V, E)$, and $\Upsilon(x) := \partial_x \log \Gamma(x)$ the Digamma function. We have

$$
\partial_{y_{kv}} [L_\beta - \log p(\beta)] = \frac{1}{\lambda^2} \sum_{v' \in N(v)} \phi_{v,v'} [y_{kv} - y_{kv'}] + \beta_{kv} \Big( \Upsilon(\bar{\beta}_k + n_k^{\mathrm{K}}) - \Upsilon(\bar{\beta}_k) +
$$

$$
+ \left\{n_{kv}^{\mathrm{KV}} > 0\right\} \left[\Upsilon(\beta_{kv}) - \Upsilon(\beta_{kv} + n_{kv}^{\mathrm{KV}})\right] \Big).
$$

The gradient with respect to $y_k$ is analogous.

# 4 Experiments

To demonstrate the usefulness of our model we applied it to a multi-lingual document collection, where we can show a substantial improvement over the standard LDA model on the coordination between topics of different languages.

## 4.1 Dataset

Since our goal is to compare topic distributions on different languages we used a parallel corpus [11] with the proceedings of the European Parliament in 11 languages. We focused on two language pairs: English/French and English/Portuguese.

Note that a parallel corpus is *not* necessary for the application of the proposed model – it is being used here only because it allows us to properly evaluate the effectiveness of our model.[3]

We treated the transcript of each speaker in each session as a document, since different speakers usually talk about different topics. We randomly sampled 1000 documents from each language, removed infrequent[4] and frequent[5] words and kept only the documents with at least 20 words. Finally, we removed all documents that lost their corresponding translations in this process. After this preprocessing we were left with 2415 documents, 805 in each language, and a vocabulary size of 23883 words.

## 4.2 Baselines

We compared our model to standard LDA, learning $\alpha$ and $\beta$, both asymmetric[6].

## 4.3 Prior

We imposed the graph based prior mentioned in Section 2.2. To build our similarity graph we used the English-French and English-Portuguese dictionaries from `http://wiki.webz.cz/dict/`, augmented with translations from Google Translate for the most frequent words in our dataset. As described earlier, each word corresponds to a vertex, with an edge[7] whenever two words match in the dictionary.

In our model $\beta = \exp(y_{kv} + y_v)$, so we want to keep both $y_{kv}$ and $y_v$ reasonably low to avoid numerical problems, as a large value of either would lead to overflows. We ensure that by setting $\lambda$, the standard deviation of their prior, fixed to one in all experiments. We did the same for the standard LDA model, where to learn an asymmetric beta we simply removed $y_{kv}$ to get $\beta = \exp(y_v)$.

## 4.4 Methodology

In our experiments we used all the English documents and a subset of the French and Portuguese ones – this is what we have in a real application, when we try to learn a topic model from web pages: the number of pages is English is far greater than in any other language.

We compared three approaches. First, we run the standard LDA model with all documents mixed together – this is one of our baselines, which we call STD1.

Next we run our proposed model, but with a slight modification to the setup: in the first half of the iterations of the Gibbs sampler we include only English documents; in the second half we add the French and Portuguese ones to the mix.[8]

Finally, as a control experiment we run the standard LDA model in this same setting: first English documents, then all languages mixed. We call this STD2.

In all experiments we run the Gibbs sampler for a total of 3000 iterations, with the number of topics fixed to 20, and keep the last sample. After a burn-in of 500 iterations, the optimisation over the word smoothing coefficients is done every 100 iterations, using an off-the-shelf L-BFGS [12] optimizer.[9]. We repeat every experiment 5 times with different randomisations.

## 4.5 Evaluation

Evaluation of topic models is an open problem – recent work [7] suggests that popular measures based on held-out likelihood, such as perplexity, do *not* capture whether topics are coherent or not. Furthermore, we need a set of measures that can assess whether or not we improved over the standard LDA model w.r.t. our goal – *to synchronize topics across different languages* – and there's no reason to believe that likelihood measures would assess that: a model where topics are synchronized across languages is not necessarily more likely than a model that is not synchronized. Therefore, to evaluate our model we compare the topic distributions of each English document with its corresponding French pair (and analogously for the other combinations: English/Portuguese and French/Portuguese), with these metrics:

**Mean $\ell_2$ distance:**

$$\frac{1}{|L_1|} \sum_{d_1 \in L_1, d_2 = F(d_1)} \left( \sum_{k=1}^{K} \left( \theta_k^{d_1} - \theta_k^{d_2} \right)^2 \right)^{\frac{1}{2}}$$

where $L_1$ denotes the set of documents in the first language, $F$ a mapping from a document in the first language to its corresponding translation in the second language and $\theta^d$ the topic distribution of document $d$.

**Mean Hellinger distance:** $\frac{1}{|L_1|} \sum_{d_1 \in L_1, d_2 = F(d_1)} \sum_{k=1}^{K} \left( \sqrt{\theta_k^{d_1}} - \sqrt{\theta_k^{d_2}} \right)^2$

**Agreements on first topic:** $\frac{1}{|L_1|} \sum_{d_1 \in L_1, d_2 = F(d_1)} I(\text{argmax}_k \theta_k^{d_1}, \text{argmax}_k \theta_k^{d_2}))$

where $I$ is the indicator function – that is, the proportion of document pairs where the most likely topic is the same for both languages.

**Mean number of agreements in top 5 topics:** $\frac{1}{|L_1|} \sum_{d_1 \in L_1, d_2 = F(d_1)} agreements(d_1, d_2)$

where $agreements(d_1, d_2)$ is the cardinality of the intersection of the 5 most likely topics of $d_1$ and $d_2$.

## 4.6 Results

In Figure 3 we compare our method (DC) to the standard LDA model (STD1 and STD2, see section 4.4), for the English-French pair[10]. In all metrics our proposed model shows a substantial improvement over the standard LDA model.

In Figures 4 and 5 we do the same for the English-Portuguese and Portuguese-French pairs, respectively, with similar results. Note that we did *not* use a Portuguese-French dictionary in any experiment.

In Figure 6 we plot the word smoothing prior for the English word *democracy* and its French and Portuguese translations, *démocratie* and *democracia*, for both the standard LDA model (STD1) and our model (DC), with 20% of the French and Portuguese documents used in training. In STD1 we don't have topic-specific priors (hence the horizontal line) and the word *democracy* has a much higher prior, because it happens more often in the dataset (since we have all English documents and only 20% of the French and Portuguese ones). In DC, however, the priors are topic-specific and quite similar, as this is enforced by the similarity graph.

To emphasize that we do not need a parallel corpus we ran a second experiment where we selected the same number of documents of each language, but assuring that for each document its corresponding translations are *not* in the dataset, and trained our model (DC) with 100 topics. This could be done with any multilingual corpus, since no parallelization is required. In this case, however, we cannot compute the distance metrics as before, since we have no information on the actual topic distributions of the documents. The best we can hope to do is to visually inspect the most likely words for the learned topics. This is shown in Table 1, for some selected topics, where the synchronization amongst the different languages is clear.

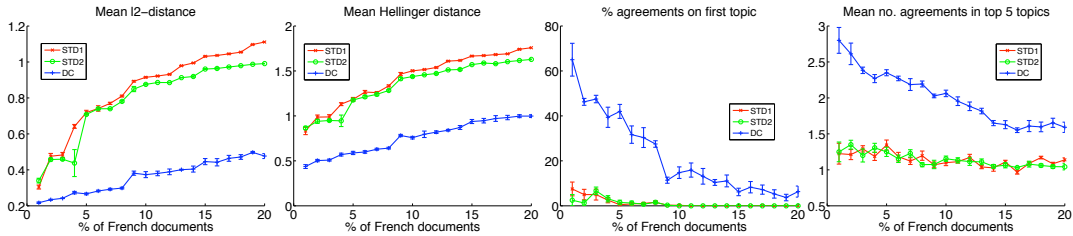

Figure 3: Comparison of topic distributions in English and French documents. See text for details.

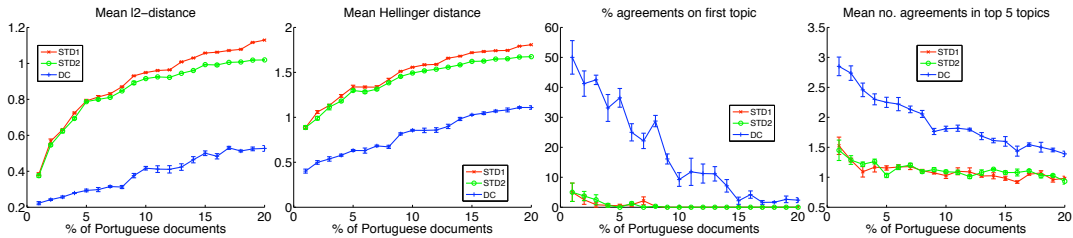

Figure 4: Comparison of topic distributions in English and Portuguese documents. See text.

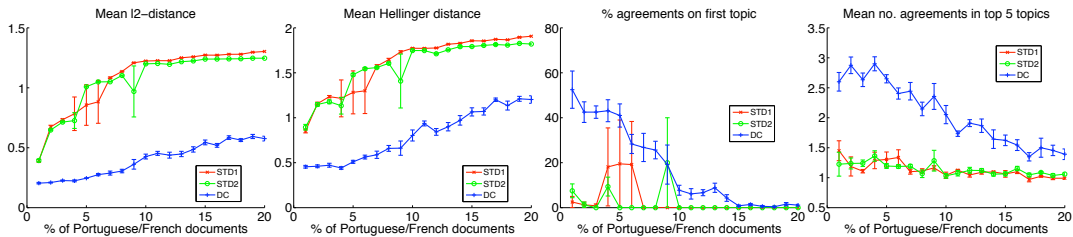

Figure 5: Comparison of topic distributions in Portuguese and French documents. See text.

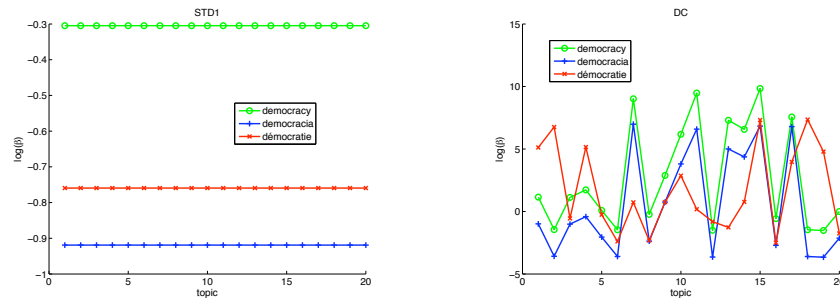

Figure 6: Word smoothing prior for two words in the standard LDA and in our model. The x-axis is the index to the topic. See text for details.

## 5 Extensions: Other Features

Although we have implemented a specific type of feature encoding for the words, our model admits a large range of applications through a suitable choice of features. In the following we discuss a number of them in greater detail.

Table 1: Top 10 words for some of the learned topics (from top to bottom, respectively, topics 8, 17, 20, 32, 49). Words are colored according to their language – English, Portuguese or French – except when ambiguous (e.g., *information* is a word in both French and English). See text for details.

| |
|---|
| amendments, alterações, amendment, amendements, alteração, use, substances, règlement, l'amendement, accept |
| élections, electoral, elections, députés, eleições, partis, proportional, eleitoral, transnational, scrutin |
| informação, information, regiões, société, l'information, acesso, aeroplanes, prix, régions, comunicação |
| stability, coordination, estabilidade, central, coordenação, plans, objectivo, stabilité, ue, list |
| monnaie, consumers, consumidores, consommateurs, l'euro, crois, s'agit, moeda, pouvoir, currency |

## 5.1 Single Language

**Distributional Similarity:** The basic idea is that words are similar if they occur in a similar context [16]. Hence, one could build a graph as outlined in Section 2.2 with edges only between words which exceed a level of proximity.

**Lexical Similarity:** For interpolation between words one could use a distribution over substrings of a word as the feature map. This is essentially what is proposed by [18]. Such lexical similarity makes the sampler less sensitive to issues such as stemming: after all, two words which reduce to the same stem will also have a high lexical similarity score, hence the estimated $\beta_{kv}$ will yield very similar topic assignments.

**Synonyms and Thesauri:** Given a list of synonyms it is reasonable to assume that they belong to related topics. This can be achieved by adding edges between a word and all of its synonyms. Since in our framework we only use this information to shape a *prior*, errors in the synonym list and multiple meanings of a word will not prove fatal.

## 5.2 Multiple Languages

**Lexical Similarity:** Similar considerations apply for inter-lingual topic models. It is reasonable to assume that lexical similarity generally points to similarity in meaning. Using such features should allow one to synchronise topics even in the absence of dictionaries. However, it is important that similarities are not hardcoded but only imposed as a prior on the topic distribution (e.g., 'gift' has different meanings in English and German).

# 6 Discussion

In this paper we described a simple yet general formalism for incorporating word features into LDA, which among other things allows us to synchronise topics across different languages. We performed a number of experiments in the multiple-language setting, in which the goal was to show that our model is able to incorporate dictionary information in order to improve topic alignment across different languages. Our experimental results reveal substantial improvement over the LDA model in the quality of topic alignment, as measured by several metrics, and in particular we obtain much improved topic alignment even across languages for which a dictionary is not used (as described in the Portuguese/French plots, see Figure 5). We also showed that the algorithm is quite effective even in the *absence* of documents that are explicitly denoted as being aligned (see Table 1). This sets it apart from [13], which requires that a significant fraction (at least 25%) of documents are paired up. Also, the model is not limited to lexical features. Instead, we could for instance also exploit syntactical information such as parse trees. For instance, noun / verb disambiguation or named entity recognition are all useful in determining the meaning of words and therefore it is quite likely that they will also aid in obtaining an improved topical mixture model.

# Acknowledgements

NICTA is funded by the Australian Government as represented by the Department of Broadband, Communications and the Digital Economy and the Australian Research Council through the ICT Centre of Excellence program.

## Footnotes

[1]Note that this is not entirely correct — the model changes slightly during one resampling pass, hence the log-likelihood that we compute is effectively the averaged log-likelihood due to an ongoing sampler. For a correct computation we would need to perform one pass through the data without resampling. Since this is wasteful, we choose the approximation instead.

[2]Here $z_i$ denotes the topic of word $i$, and $z_{\neg i}$ the topics of all words in the corpus except for $i$.

[3]To emphasise this point, later in this section we show experiments with non-parallel corpora, in which case we have to rely on visual inspection to assess the outcomes.

[4]Words that occurred less than 3 times in the corpus.

[5]Words that occurred more than $M/10$ times in the corpus, where $M$ is the total number of documents.

[6]That is, we don't assume all coordinates of $\alpha$ and $\beta$ are identical.

[7]All edges have a fixed weight of one in this case.

[8]We need to start with only one language so that an initial topic-word distribution is built; once that is done the priors are learned and can be used to guide the topic-word distributions in other languages.

[9]http://www.chokkan.org/software/liblbfgs

[10]See the Appendix for run times.

# References

[1] David Andrzejewski, Xiaojin Zhu, and Mark Craven. Incorporating domain knowledge into topic modeling via Dirichlet Forest priors. In *ICML*, pages 1–8. ACM Press, 2009.

[2] C. Antoniak. Mixtures of Dirichlet processes with applications to Bayesian nonparametric problems. *Annals of Statistics*, 2:1152–1174, 1974.

[3] David M. Blei and John D. Lafferty. Dynamic topic models. In W. W. Cohen and A. Moore, editors, *ICML*, volume 148, pages 113–120. ACM, 2006.

[4] David M. Blei, Andrew Y. Ng, and Michael I. Jordan. Latent Dirichlet allocation. *Journal of Machine Learning Research*, 3:993–1022, January 2003.

[5] Jordan Boyd-Graber, David Blei, and Xiaojin Zhu. A Topic Model for Word Sense Disambiguation. In *EMNLP-CoNLL*, pages 1024–1033, 2007.

[6] Jordan Boyd-Graber and David M. Blei. Multilingual topic models for unaligned text. In *Proceedings of the 25th Conference in Uncertainty in Artificial Intelligence (UAI)*, 2009.

[7] Jonathan Chang, Jordan Boyd-Graber, Sean Gerrish, Chong Wang, and David Blei. Reading tea leaves: How humans interpret topic models. In Y. Bengio, D. Schuurmans, J. Lafferty, C. K. I. Williams, and A. Culotta, editors, *NIPS*, pages 288–296. 2009.

[8] Thomas L. Griffiths and Mark Steyvers. Finding scientific topics. *Proceedings of the National Academy of Sciences*, 101:5228–5235, 2004.

[9] Woosung Kim and Sanjeev Khudanpur. Lexical triggers and latent semantic analysis for crosslingual language model adaptation. *ACM Transactions on Asian Language Information Processing*, 3, 2004.

[10] T.B. Kirkpatrick, A.B. Côté, J. DeNero, and Dan Klein. Painless Unsupervised Learning with Features. In *Human Language Technologies: The 2010 Annual Conference of the North American Chapter of the Association for Computational Linguistics*, 2010.

[11] Philipp Koehn. Europarl: A parallel corpus for statistical machine translation. In *Machine Translation Summit X*, pages 79–86, 2005.

[12] Dong C. Liu and Jorge Nocedal. On the limited memory BFGS method for large scale optimization. *Mathematical Programming*, 45(3):503–528, 1989.

[13] David Mimno, Hanna M. Wallach, Jason Naradowsky, David A. Smith, and Andrew McCallum. Polylingual topic models. In *Proceedings of the 2009 Conference on Empirical Methods in Natural Language Processing*, pages 880–889, Singapore, August 2009. ACL.

[14] David M. Mimno and Andrew McCallum. Topic models conditioned on arbitrary features with dirichlet-multinomial regression. In D. A. McAllester and P. Myllymäki, editors, *UAI, Proceedings of the 24th Conference in Uncertainty in Artificial Intelligence*, pages 411–418. AUAI Press, 2008.

[15] Xiaochuan Ni, Jian-Tao Sun, Jian Hu, and Zheng Chen. Mining multilingual topics from wikipedia. In *18th International World Wide Web Conference*, pages 1155–1155, April 2009.

[16] Patrick Pantel and Dekang Lin. Discovering word senses from text. In David Hand, Daniel Keim, and Raymond Ng, editors, *Proceedings of the Eighth ACM SIGKDD International Conference on Knowledge Discovery and Data Mining*, pages 613–619, New York, July 2002. ACM Press.

[17] Noah A Smith and Shay B Cohen. The Shared Logistic Normal Distribution for Grammar Induction. In *NIPS Workshop on Speech and Language: Unsupervised Latent-Variable Models,*, pages 1–4, 2008.

[18] S. V. N. Vishwanathan and A. J. Smola. Fast kernels for string and tree matching. In S. Becker, S. Thrun, and K. Obermayer, editors, *Advances in Neural Information Processing Systems 15*, pages 569–576. MIT Press, Cambridge, MA, 2003.

[19] Limin Yao, David Mimno, and Andrew McCallum. Efficient methods for topic model inference on streaming document collections. In *KDD'09*, 2009.

[20] Bing Zhao and Eric P. Xing. BiTAM: Bilingual Topic AdMixture Models for Word Alignment. In *In Proceedings of the 44th Annual Meeting of the Association for Computational Linguistics (ACL'06)*, 2006.

